# Directional-Unit Boltzmann Machines

**Richard S. Zemel**
Computer Science Dept.
University of Toronto
Toronto, ONT M5S 1A4

**Christopher K. I. Williams**
Computer Science Dept.
University of Toronto
Toronto, ONT M5S 1A4

**Michael C. Mozer**
Computer Science Dept.
University of Colorado
Boulder, CO 80309–0430

## Abstract

We present a general formulation for a network of stochastic directional units. This formulation is an extension of the Boltzmann machine in which the units are not binary, but take on values in a cyclic range, between 0 and $2\pi$ radians. The state of each unit in a Directional-Unit Boltzmann Machine (DUBM) is described by a complex variable, where the phase component specifies a direction; the weights are also complex variables. We associate a quadratic energy function, and corresponding probability, with each DUBM configuration. The conditional distribution of a unit's stochastic state is a circular version of the Gaussian probability distribution, known as the von Mises distribution. In a mean-field approximation to a stochastic DUBM, the phase component of a unit's state represents its mean direction, and the magnitude component specifies the degree of certainty associated with this direction. This combination of a value and a certainty provides additional representational power in a unit. We describe a learning algorithm and simulations that demonstrate a mean-field DUBM's ability to learn interesting mappings.

Many kinds of information can naturally be represented in terms of angular, or directional, variables. A circular range forms a suitable representation for explicitly directional information, such as wind direction, as well as for information where the underlying range is periodic, such as days of the week or months of the year. In computer vision, tangent fields and optic flow fields are represented as fields of oriented line segments, each of which can be described by a magnitude and direction. Directions can also be used to represent a set of symbolic labels, e.g., object label $A$ at 0, and object label $B$ at $\pi/2$ radians. We discuss below some advantages of representing symbolic labels with directional units.

These and many other phenomena can be usefully encoded using a *directional* representation—a polar coordinate representation of complex values in which the phase parameter indicates a direction between 0 and $2\pi$ radians. We have devised a general formulation of networks of stochastic directional units. This paper describes a *directional-unit Boltzmann machine* (DUBM), which is a novel generalization of a Boltzmann machine (Ackley, Hinton and Sejnowski, 1985) in which the units are not binary, but instead take on directional values between 0 and $2\pi$.

## 1  STOCHASTIC DUBM

A stochastic directional unit takes on values on the unit circle. We associate with unit $j$ a random variable $Z_j$; a particular state of $j$ is described by a complex number with magnitude one and direction, or phase $\tau_j$: $z_j = e^{i\tau_j}$ .

The weights of a DUBM also take on complex values. The weight from unit $k$ to unit $j$ is: $w_{jk} \equiv b_{jk}e^{i\theta_{jk}}$. We constrain the weight matrix $W$ to be Hermitian: $W^T = W^*$, where the diagonal elements of the matrix are zero, and the asterisk indicates the complex conjugate operation. Note that if the components are real, then $\mathbf{W}^T = \mathbf{W}$, which is a real symmetric matrix. Thus, the Hermitian form is a natural generalization of weight symmetry to the complex domain.

This definition of $\mathbf{W}$ leads to a Hermitian quadratic form that generalizes the real quadratic form of the Hopfield energy function:

$$E(\mathbf{z}) = -1/2\, \mathbf{z}^{*T}\mathbf{W}\mathbf{z} = -1/2 \sum_{j,k} z_j z_k^* w_{jk} \tag{1}$$

where $\mathbf{z}$ is the vector of the units' complex states in a particular global configuration. Noest (1988) independently proposed this energy function. It is similar to that used in Fradkin, Huberman, and Shenker's (1978) generalization of the XY model of statistical mechanics to allow arbitary weight phases $\theta_{jk}$, and coupled oscillator models, e.g., Baldi and Meir (1990).

We can define a probability distribution over the possible states of a stochastic network using the Boltzmann factor. In a DUBM, we can describe the energy as a function of the state of a particular unit $j$:

$$E(Z_j = z_j) = -1/2\left[\sum_k z_j z_k^* w_{jk} + \sum_k z_k z_j^* w_{kj}\right]$$

We define

$$x_j = \sum_k z_k w_{jk}^* = a_j e^{i\alpha_j}$$

to be the net input to unit $j$, where $a_j$ and $\alpha_j$ denote the magnitude and phase of $x_j$, respectively.

Applying the Boltzmann factor, we find that the probability that unit $j$ is in a particular state is proportional to:

$$p(Z_j = z_j) \propto e^{-\beta E(Z_j = z_j)} = e^{\beta a_j \cos(\tau_j - \alpha_j)} \tag{2}$$

where $\beta$ is the reciprocal of the system temperature.

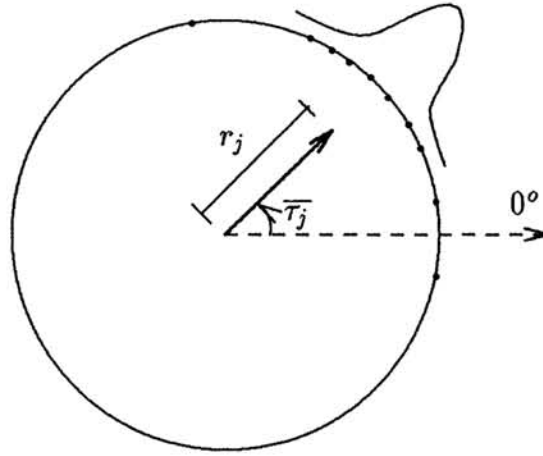

Figure 1: A circular normal density function laid over a unit circle. The dots along the circle represent samples of the circular normal random variable $Z_j$. The expected direction of $Z_j$, $\overline{\tau_j}$, is $\pi/4$; $r_j$ is its resultant length.

This probability distribution for a unit's state corresponds to a distribution known as the *von Mises*, or *circular normal*, distribution (Mardia, 1972). Two parameters completely characterize this distribution: a mean direction $\overline{\tau} = (0, 2\pi]$ and a concentration parameter $m > 0$ that behaves like the reciprocal of the variance of a Gaussian distribution on a linear random variable. The probability density function of a circular normal random variable $Z$ is[1]:

$$p(\tau; \ \overline{\tau}, m) \ = \ \frac{1}{2\pi I_0(m)} \ e^{m\cos(\tau - \overline{\tau})} \tag{3}$$

From Equations 2 and 3, we see that if a unit adopts states according to its contribution to the system energy, it will be a circular normal variable with mean direction $\alpha_j$ and concentration parameter $m_j \ = \ \beta a_j$. These parameters are directly determined by the net input to the unit.

Figure 1 shows a circular normal density function for $Z_j$, the state of unit $j$. This figure also shows the expected value of its stochastic state, which we define as:

$$y_j \ = \ < Z_j > = \ r_j e^{i\gamma_j} \tag{4}$$

where $\gamma_j$, the phase of $y_j$, is the mean direction and $r_j$, the magnitude of $y_j$, is the *resultant length*. For a circular normal random variable, $\gamma_j = \overline{\tau_j}$, and $r_j \ = \ \frac{I_1(m_j)}{I_0(m_j)}$.[2]

When samples of $Z_j$ are concentrated on a small arc about the mean (see Figure 1), $r_j$ will approach length one. This corresponds to a large concentration parameter $(m_j \ = \ \beta a_j)$. Conversely, for small $m_j$, the distribution approaches the uniform distribution on the circle, and the resultant length falls toward zero. For a uniform distribution, $r_j = 0$. Note that the concentration parameter for a unit's circular

normal density function is proportional to $\beta$, the reciprocal of the system temperature. Higher temperatures will thus have the effect of making this distribution more uniform, just as they do in a binary-unit Boltzmann machine.

## 2    EMERGENT PROPERTIES OF A DUBM

A network of directional units as defined above contains two important emergent properties. The first property is that the magnitude of the net input to unit $j$ describes the extent to which its various inputs "agree". Intuitively, one can think of each component $z_k w_{jk}^*$ of the sum that comprises $x_j$ as predicting a phase for unit $j$. When the phases of these components are equal, the magnitude of $x_j$, $a_j$, is maximized. If these phase predictions are far apart, then they will act to cancel each other out, and produce a small $a_j$. Given $x_j$, we can compute the expected value of the output of unit $j$. The expected direction of the unit roughly represents the weighted average of the phase predictions, while the resultant length is a monotonic function of $a_j$ and hence describes the agreement between the various predictions.

The key idea here is that the resultant length directly describes the degree of certainty in the expected direction of unit $j$. Thus, a DUBM naturally incorporates a representation of the system's confidence in a value. This ability to combine several sources of evidence, and not only represent a value but also describe the certainty of that value is an important property that may be useful in a variety of domains.

The second emergent property is that the DUBM energy is globally *rotation-invariant*—$E$ is unaffected when the same rotation is applied to all units' states in the network. For each DUBM configuration, there is an equivalence class of configurations which have the same energy. In a similar way, we find that the magnitude of $x_j$ is *rotation-invariant*. That is, when we translate the phases of all units but one by some phase, the magnitude of that unit is unaffected. This property underlies one of the key advantages of the representation: both the magnitude of a unit's state as well as system energy depend on the *relative* rather than *absolute* phases of the units.

## 3    DETERMINISTIC DUBM

Just as in deterministic binary-unit Boltzmann machines (Peterson and Anderson, 1987; Hinton, 1989), we can greatly reduce the computational time required to run a large stochastic system if we invoke the *mean-field approximation*, which states that once the system has reached equilibrium, the stochastic variables can be approximated by their mean values. In this approximation, the variables are treated as independent, and the system probability distribution is simply the product of the probability distributions for the individual units.

Gislén, Peterson, and Söderberg (1992) originally proposed a mean-field theory for networks of directional (or "rotor") units, but only considered the case of real-valued weights. They derived the mean-field consistency equations by using the saddle-point method. Our approach provides an alternative, perhaps more intuitive derivation, due to the use of the circular normal distribution.

We can directly describe these mean values based on the circular normal interpretation. We still denote the net input to a unit $j$ as $x_j$:

$$x_j = \sum_k y_k w^*_{jk} = a_j e^{i\alpha_j} \tag{5}$$

Once equilibrium has been reached, the state of unit $j$ is $y_j$, the expected value of $Z_j$ given the mean-field approximation:

$$y_j = r_j e^{i\gamma_j} = \frac{I_1(\beta a_j)}{I_0(\beta a_j)} e^{i\alpha_j} \tag{6}$$

In the stochastic as well as the deterministic system, units evolve to minimize the *free energy*, $F = <E> - TH$. The calculation of $H$, the entropy of the system, follows directly from the circular normal distribution and the mean-field approximation. We can derive mean-field consistency equations for $x_j$ and $y_j$ by minimizing the mean-field free energy, $F_{MF}$, with respect to each variable independently. The resulting equations match the mean-field equations (Equations 5 and 6) derived directly from the circular normal probability density function. They also match the special case derived by Gislén *et al.* for real-valued weights.

We have implemented a DUBM using the mean-field approximation. We solve for a consistent set of **x** and **y** values by performing synchronous updates of the discrete-time approximation of the set of differential equations based on the net input to each unit $j$. We update the $x_j$ variables using the following differential equation:

$$\frac{dx_j}{dt} = -x_j + \sum_k y_k w^*_{jk} \tag{7}$$

which has Equation 5 as its steady-state solution. In the simulations, we use simulated annealing to help find good minima of $F_{MF}$.

Just as for the Hopfield binary-state network, it can be shown that the free energy always decreases during the dynamical evolution described in Equation 7 (Zemel, Williams and Mozer, 1992). The equilibrium solutions are free energy minima.

## 4   DUBM LEARNING

The units in a DUBM can be arranged in a variety of architectures. The appropriate method for determining weight values for the network depends on the particular class of network architecture. In an autoassociative network containing a single set of interconnected units, the weights can be set directly from the training patterns. If hidden units are required to perform a task, then an algorithm for learning the weights is required. We use an algorithm that generalizes the Boltzmann machine training algorithm (Ackley, Hinton and Sejnowski, 1985; Peterson and Anderson, 1987) to these networks.

As in the standard Boltzmann machine learning algorithm, the partial derivative of the objective function with respect to a weight depends on the difference between the partials of two mean-field free energies: one when both input and output units are clamped, and the other when only the input units are clamped. On a given

training case, for each of these stages we let the network settle to equilibrium and then calculate the following derivatives:

$$\partial F_{MF}/\partial b_{jk} = -r_j r_k \cos(\gamma_j - \gamma_k + \theta_{jk})$$
$$\partial F_{MF}/\partial \theta_{jk} = r_j r_k b_{jk} \sin(\gamma_j - \gamma_k + \theta_{jk})$$

The learning algorithm uses these gradients to find weight values that will minimize the objective over a training set.

## 5    EXPERIMENTAL RESULTS

We present below some illustrative examples to show that an adaptive network of directional units can be used in a range of paradigms, including associative memory, input/output mappings, and pattern completion.

### 5.1    SIMPLE AUTOASSOCIATIVE DUBM

The first set of experiments considers a simple autoassociative DUBM, which contains no hidden units, and the units are fully connected. As in a standard Hopfield network, the weights are set directly from the training patterns; they equal the superposition of the outer product of the patterns.

We have run several experiments with simple autoassociative DUBMs. The empirical results parallel those for binary-unit autoassociative networks. We find, for example, that a network containing 30 fully interconnected units is capable of reliably settling from a corrupted version of one of 4 stored patterns to a state near the pattern. These patterns thus form stable attractors, as the network can perform pattern completion and clean-up from noisy inputs. The rotation-invariance property of the energy function allows any rotated version of a training pattern to also act as an attractor. The network's performance rapidly degrades for more than 4 orthogonal patterns; the patterns themselves no longer act as fixed-points, and many random initial states end in states far from any stored pattern. In addition, more orthogonal patterns can be stored than random patterns. See Noest (1988) for an analysis of the capacity of an autoassociative DUBM with sparse and asymmetric connections.

### 5.2    LEARNING INPUT/OUTPUT MAPPINGS

We have also used the mean-field DUBM learning algorithm to learn the weights in networks containing hidden units. We have experimented with a task that is well-suited to a directional representation. There is a single-jointed robot arm, anchored at a point, as shown in Figure 2. The input consists of two angles: the angle between the first arm segment and the positive x-axis ($\lambda$), and the angle between the two arm segments ($\rho$). The two segments each have a fixed length, $A$ and $B$; these are not explicitly given to the network. The output is the angle between the line connecting the two ends of the arm and the x-axis ($\mu$). This target angle is related in a complex, non-linear way to the input angles—the network must learn to approximate the following trigonometric relationship:

$$\mu = \arctan\left(\frac{A\sin\lambda - B\sin(\lambda + \rho)}{A\cos\lambda - B\cos(\lambda + \rho)}\right)$$

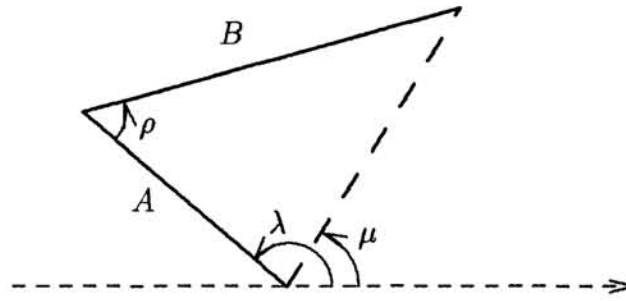

Figure 2: A sample training case for the robot arm problem. The arm consists of two fixed-length segments, $A$ and $B$, and is anchored on the $x$-axis. The two angles, $\lambda$ and $\rho$, are given as input for each case, and the target output is the angle $\mu$.

With 500 training cases, a DUBM with 2 input units and 8 hidden units is able to learn the task so that it can accurately estimate $\mu$ for novel patterns. The learning requires 200 iterations of a conjugate gradient training algorithm. On each of 100 testing patterns, the resultant length of the output unit exceeds .85, and the mean error on the angle is less than .05 radians. The network can also learn the task with as few as 5 hidden units, with a concomitant decrease in learning speed. The compact nature of this network shows that the directional units form a natural, efficient representation for this problem.

### 5.3   COMPLEX PATTERN COMPLETION

Our earlier work described a large-scale DUBM that attacks a difficult problem in computer vision: image segmentation. In MAGIC (Mozer et al., 1992), directional values are used to represent alternative labels that can be assigned to image features. The goal of MAGIC is to learn to assign appropriate object labels to a set of image features (e.g., edge segments) based on a set of examples. The idea is that the features of a given object should have consistent phases, with each object taking on its own phase. The units in the network are arranged into two layers—feature and hidden—and the computation proceeds by randomly initializing the phases of the units in the feature layer, and settling on a labeling through a relaxation procedure. The units in the hidden layer learn to detect spatially local configurations of the image features that are labeled in a consistent manner across the training examples.

MAGIC successfully learns to segment novel scenes consisting of overlapping geometric objects. The emergent DUBM properties described above are essential to MAGIC's ability to perform this task. The complex weights are necessary in MAGIC, as the weights encode statistical regularities in the relationships between image features, e.g., that two features typically belong to the same object (i.e., have similar phase values) or to different objects (i.e., are out of phase). The fact that a unit's resultant length reflects the certainty in a phase label allows the system to decide which phase labels to use when updating labels of neighboring features: the initially random phases are ignored, while confident labels are propagated. Finally, the rotation-invariance property allows the system to assign labels to features in a manner consistent with the relationships described in the weights, where it is the *relative* rather than *absolute* phases of the units that are important.

# 6   CURRENT DIRECTIONS

We are currently extending this work in a number of directions. We are extending the definition of a DUBM to combine binary and directional units (Radford Neal, personal communication). This expanded representation may be useful in domains with directional data that is not present everywhere. For example, it can be directly applied to the object labeling problem explored in MAGIC. The binary aspect of the unit can describe whether a particular image feature is present or absent. This may enable the system to handle various complications, particularly labeling across gaps along the contour of an object. Finally, we are applying a DUBM network to the interesting and challenging problem of time-series prediction of wind directions.

## Acknowledgements

The authors thank Geoffrey Hinton for his generous support and guidance. We thank Radford Neal, Peter Dayan, Conrad Galland, Sue Becker, Steve Nowlan, and other members of the Connectionist Research Group at the University of Toronto for helpful comments regarding this work. This research was supported by a grant from the Information Technology Research Centre of Ontario to Geoffrey Hinton, and NSF Presidential Young Investigator award IRI–9058450 and grant 90–21 from the James S. McDonnell Foundation to MM.

## Footnotes

[1]The normalization factor $I_0(m)$ is the modified Bessel function of the first kind and order zero. An integral representation of this function is $I_0(m) = \frac{1}{\pi} \int_0^\pi e^{\pm m\cos\theta} d\theta$. It can be computed by numerical routines.

[2]An integral representation of the modified Bessel function of the first kind and order $k$ is $I_k(m) = \frac{1}{\pi} \int_0^\pi e^{m\cos\theta} \cos(k\theta) d\theta$. Note that $I_1(m) = dI_0(m)/dm$.

## References

Ackley, D. H., Hinton, G. E., and Sejnowski, T. J. (1985). A learning algorithm for Boltzmann machines. *Cognitive Science*, 9:147–169.

Baldi, P. and Meir, R. (1990). Computing with arrays of coupled oscillators: An application to preattentive texture discrimination. *Neural Computation*, 2(4):458–471.

Fradkin, E., Huberman, B. A., and Shenker, S. H. (1978). Gauge symmetries in random magnetic systems. *Physical Review B*, 18(9):4789–4814.

Gislén, L., Peterson, C., and Söderberg, B. (1992). Rotor neurons: Basic formalism and dynamics. *Neural Computation*, 4(5):737–745.

Hinton, G. E. (1989). Deterministic Boltzmann learning performs steepest descent in weight-space. *Neural Computation*, 1(2):143–150.

Mardia, K. V. (1972). *Statistics of Directional Data*. Academic Press, London.

Mozer, M. C., Zemel, R. S., Behrmann, M., and Williams, C. K. I. (1992). Learning to segment images using dynamic feature binding. *Neural Computation*, 4(5):650–665.

Noest, A. J. (1988). Phasor neural networks. In *Neural Information Processing Systems*, pages 584–591, New York. AIP.

Peterson, C. and Anderson, J. R. (1987). A mean field theory learning algorithm for neural networks. *Complex Systems*, 1:995–1019.

Zemel, R. S., Williams, C. K. I., and Mozer, M. C. (1992). Adaptive networks of directional units. Technical Report CRG-TR-92-2, University of Toronto.